# Model Uncertainty in Classical Conditioning

**A. C. Courville**[*][1,3], **N. D. Daw**[2,3], **G. J. Gordon**[4], **and D. S. Touretzky**[2,3]
[1]Robotics Institute, [2]Computer Science Department,
[3]Center for the Neural Basis of Cognition,
[4]Center for Automated Learning and Discovery
Carnegie Mellon University, Pittsburgh, PA 15213
{aaronc,daw,ggordon,dst}@cs.cmu.edu

## Abstract

We develop a framework based on Bayesian model averaging to explain how animals cope with uncertainty about contingencies in classical conditioning experiments. Traditional accounts of conditioning fit parameters within a fixed generative model of reinforcer delivery; uncertainty over the model structure is not considered. We apply the theory to explain the puzzling relationship between second-order conditioning and conditioned inhibition, two similar conditioning regimes that nonetheless result in strongly divergent behavioral outcomes. According to the theory, second-order conditioning results when limited experience leads animals to prefer a simpler world model that produces spurious correlations; conditioned inhibition results when a more complex model is justified by additional experience.

## 1   Introduction

Most theories of classical conditioning, exemplified by the classic model of Rescorla and Wagner [7], are wholly concerned with *parameter learning*. They assume a fixed (often implicit) generative model $m$ of reinforcer delivery and treat conditioning as a process of estimating values for the parameters $\mathbf{w}_m$ of that model. Typically, these parameters represent the rates of reinforcers delivered in the presence of various stimuli. Using the model and the parameters, the probability of reinforcer delivery can be estimated; such estimates are assumed to give rise to conditioned responses in behavioral experiments. More overtly statistical theories have treated *uncertainty* in the parameter estimates, which can influence predictions and learning [4].

In realistic situations, the underlying contingencies of the environment are complex and unobservable, and it can thus make sense to view the model $m$ as itself uncertain and subject to learning, though (to our knowledge) no explicitly statistical theories of conditioning have yet done so. Under the standard Bayesian approach, such uncertainty can be treated analogously to parameter uncertainty, by representing knowledge about $m$ as a distribution over a set of possible models, conditioned on evidence. Here we advance this idea as a high-level computational framework for the role of model learning in classical conditioning. We do not concentrate on how the brain might implement these processes, but rather explore the behavior that a system approximating Bayesian reasoning should exhibit. This work

establishes a relationship between theories of animal learning and a recent line of theory by Tenenbaum and collaborators, which uses similar ideas about Bayesian model learning to explain human causal reasoning [9].

We have applied our theory to a variety of standard results in animal conditioning, including acquisition, negative and positive patterning, and forward and backward blocking. Here we present one of the most interesting and novel applications, an explanation of a rather mysterious classical conditioning phenomenon in which opposite predictions about the likelihood of reinforcement can arise from different amounts of otherwise identical experience [11]. The opposing effects, both well known, are called second-order conditioning and conditioned inhibition. The theory explains the phenomenon as resulting from a tradeoff between evidence and model complexity.

## 2   A Model of Classical Conditioning

In a conditioning trial, a set of conditioned stimuli $CS \equiv \{A, B, \dots\}$ is presented, potentially accompanied by an unconditioned stimulus or reinforcement signal, $US$. We represent the $j$th stimulus with a binary random variable $y_j$ such that $y_j = 1$ when the stimulus is present. Here the index $j$, $1 \leq j \leq s$, ranges over both the $(s-1)$ conditioned stimuli and the unconditioned stimulus. The collection of trials within an experimental protocol constitutes a training data set, $\mathcal{D} = \{y_{jt}\}$, indexed by stimulus $j$ and trial $t$, $1 \leq t \leq T$.

We take the perspective that animals are attempting to recover the generative process underlying the observed stimuli. We claim they assert the existence of latent causes, represented by the binary variables $x_i \in \{0, 1\}$, responsible for evoking the observed stimuli. The relationship between the latent causes and observed stimuli is encoded with a sigmoid belief network. This particular class of models is not essential to our conclusions; many model classes should result in similar behavior.

**Sigmoid Belief Networks**   In sigmoid belief networks, local conditional probabilities are defined as functions of weighted sums of parent nodes. Using our notation,

$$P(y_j = 1 \mid x_1, \dots, x_c, \mathbf{w}_m, m) = (1 + \exp(-\sum_i w_{ij} x_i - w_{y_j}))^{-1}, \qquad (1)$$

and $P(y_j = 0 \mid x_1, \dots, x_c, \mathbf{w}_m, m) = 1 - P(y_j = 1 \mid x_1, \dots, x_c, \mathbf{w}_m, m)$. The weight, $w_{ij}$, represents the influence of the parent node $x_i$ on the child node $y_j$. The bias term $w_{y_j}$ encodes the probability of $y_j$ in the absence of all parent nodes. The parameter vector $\mathbf{w}_m$ contains all model parameters for model structure $m$.

The form of the sigmoid belief networks we consider is represented as a directed graphical model in Figure 1a, with the latent causes as parents of the observed stimuli. The latent causes encode the intratrial correlations between stimuli — we do not model the temporal structure of events within a trial. Conditioned on the latent causes, the stimuli are mutually independent. We can express the conditional joint probability of the observed stimuli as $\prod_{j=1}^{s} P(y_j \mid x_1, \dots, x_c, \mathbf{w}_m, m)$.

Similarly, we assume that trials are drawn from a stationary process. We do not consider trial order effects, and we assume all trials are mutually independent. (Because of these simplifying assumptions, the present model cannot address a number of phenomena such as the difference between latent inhibition, partial reinforcement, and extinction.) The resulting likelihood function of the training data, with latent causes marginalized, is:

$$P(\mathcal{D} \mid \mathbf{w}_m, m) = \prod_{t=1}^{T} \sum_{\mathbf{x}} \prod_{j=1}^{s} P(y_{jt} \mid \mathbf{x}, \mathbf{w}_m, m) P(\mathbf{x} \mid \mathbf{w}_m, m), \qquad (2)$$

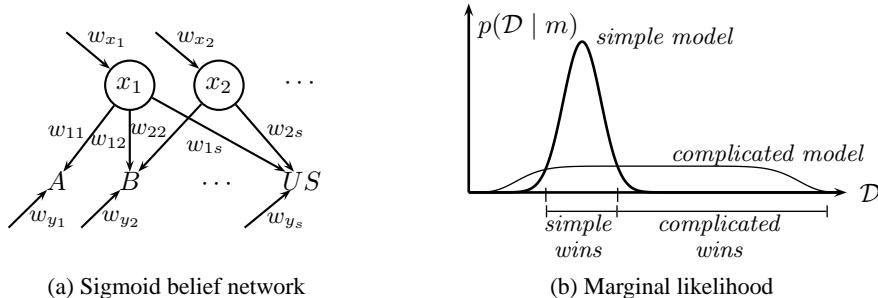

(a) Sigmoid belief network  (b) Marginal likelihood

Figure 1: (a) An example from the proposed set of models. Conditional dependencies are depicted as links between the latent causes ($x_1$, $x_2$) and the observed stimuli ($A$, $B$, $US$) during a trial. (b) Marginal likelihood of the data, $\mathcal{D}$, for a simple model and a more complicated model (after MacKay [5]).

where the sum is over all combinations of values of $\mathbf{x} = [x_1, \ldots, x_c]$ and $P(\mathbf{x} \mid \mathbf{w}_m, m) = \prod_{i=1}^{c} (1 + \exp(-1^{x_i} w_{x_i}))^{-1}$.

Sigmoid belief networks have a number of appealing properties for modeling conditioning. First, the sigmoid belief network is capable of compactly representing correlations between groups of observable stimuli. Without a latent cause, the number of parameters required to represent these correlations would scale exponentially with the number of stimuli. Second, the parent nodes, interacting additively, constitute a factored representation of state. This is advantageous as it permits generalization to novel combinations of factors. Such additivity has frequently been observed in conditioning experiments [7].

## 2.1 Prediction under Parameter Uncertainty

Consider a particular network structure, $m$, with parameters $\mathbf{w}_m$. Given $m$ and a set of trials, $\mathcal{D}$, the uncertainty associated with the choice of parameters is represented in a posterior distribution over $\mathbf{w}_m$. This posterior is given by Bayes' rule, $p(\mathbf{w}_m \mid \mathcal{D}, m) \propto P(\mathcal{D} \mid \mathbf{w}_m, m) p(\mathbf{w}_m \mid m)$, where $P(D \mid m)$ is from Equation 2 and $p(\mathbf{w}_m \mid m)$ is the prior distribution over the parameters of $m$. We assume the model parameters are *a priori* independent. $p(\mathbf{w}_m \mid m) = \prod_{ij} p(w_{ij}) \prod_i p(w_{x_i}) \prod_j p(w_{y_j})$, with Gaussian priors for weights $p(w_{ij}) = \mathcal{N}(0, 3)$, latent cause biases $p(w_{x_i}) = \mathcal{N}(0, 3)$, and stimulus biases $p(w_{y_j}) = \mathcal{N}(-15, 1)$, the latter reflecting an assumption that stimuli are rare in the absence of causes.

In conditioning, the test trial measures the conditioned response ($CR$). This is taken to be a measure of the animal's estimate of the probability of reinforcement conditioned on the present conditioned stimuli $CS$. This probability is also conditioned on the absence of the remaining stimuli; however, in the interest of clarity, our notation suppresses these absent stimuli. In the Bayesian framework, given $m$, this probability, $P(US \mid CS, m, \mathcal{D})$ is determined by integrating over all values of the parameters weighted by their posterior probability density,

$$P(US \mid CS, m, \mathcal{D}) = \int P(US \mid CS, \mathbf{w}_m, m, \mathcal{D}) p(\mathbf{w}_m \mid m, \mathcal{D}) \, d\mathbf{w}_m \qquad (3)$$

## 2.2 Prediction under Model Uncertainty

In the face of uncertainty about which is the correct model of contingencies in the world —
for instance, whether a reinforcer is independent of a tone stimulus — a standard Bayesian
approach is to marginalize out the influence of the model choice,

$$P(US \mid CS, \mathcal{D}) = \sum_m P(US \mid CS, m, \mathcal{D}) P(m \mid \mathcal{D}) \tag{4}$$

$$= \sum_m \int P(US \mid CS, \mathbf{w}_m, m, \mathcal{D}) p(\mathbf{w}_m \mid m, \mathcal{D}) P(m \mid \mathcal{D}) \, d\mathbf{w}_m$$

The posterior over models, $p(m \mid \mathcal{D})$, is given by:

$$P(m \mid \mathcal{D}) = \frac{P(\mathcal{D} \mid m) P(m)}{\sum_{m'} P(\mathcal{D} \mid m') P(m')}, \quad P(\mathcal{D} \mid m) = \int P(\mathcal{D} \mid \mathbf{w}_m, m) p(\mathbf{w}_m \mid m) \, d\mathbf{w}_m$$

The marginal likelihood $P(\mathcal{D} \mid m)$ is the probability of the data under model $m$, marginal-
izing out the model parameters. The marginal likelihood famously confers an automatic
Occam's razor effect on the average of Equation 4. Under complex models, parameters
can be found to boost the probability of particular data sets that would be unlikely under
simpler models, but any particular parameter choice is also less likely in more complex
models. Thus there is a tradeoff between model fidelity and complexity (Figure 1b).

We also encode a further preference for simpler models through the prior over model struc-
ture, which we factor as $P(m) = P(c) \prod_{i=1}^c P(l_i)$, where $c$ is the number of latent causes
and $l_i$ is the number of directed links emanating from $x_i$. The priors over $c$ and $l_i$ are in
turn given by,

$$P(c) = \begin{cases} \frac{10^{-3c}}{\sum_{c'=0}^5 10^{-3c'}} & \text{if } 0 \le c \le 5 \\ 0 & \text{otherwise} \end{cases} \quad \text{and} \quad P(l_i) = \begin{cases} \frac{10^{-3l_i}}{\sum_{l_i'=0}^s 10^{-3l_i'}} & \text{if } 0 \le l_i \le 4 \\ 0 & \text{otherwise} \end{cases}$$

In the Bayesian model average, we consider the set of sigmoid belief networks with a
maximum of 4 stimuli and 5 latent causes.

This strong prior over model structures is required in addition to the automatic Occam's
razor effect in order to explain the animal behaviors we consider. This probably is due to
the extreme abstraction of our setting. With generative models that included, e.g., temporal
ordering effects and multiple perceptual dimensions, model shifts equivalent to the addition
of a single latent variable in our setting would introduce a great deal of additional model
complexity and require proportionally more evidential justification.

## 2.3 Monte Carlo Integration

In order to determine the predictive probability of reinforcement, Bayesian model aver-
aging requires that we evaluate Equation 4. Unfortunately, the integral is not amenable
to analytic solution. Hence we approximate the integral with a sum over samples from
the posterior $p(\mathbf{w}_m, m \mid \mathcal{D})$. Acquiring samples is complicated by the need to sample
over parameter spaces of different dimensions. In the simulations reported here, we solved
this problem and obtained samples using a reversible jump Markov chain Monte Carlo
(MCMC) method [2]. A new sample in the chain is obtained by proposing perturbations
to the current sample's model structure or parameters.[1] Jumps include the addition or re-
moval of links or latent causes, or updates to the stimulus biases or weights. To improve
mixing over the different modes of the target distribution, we used exchange MCMC, which
enables fast mixing between modes through the coupling of parallel Markov chains [3].

| Group | $A$-$US$ | $A$-$X$ | $B$-$US$ | Test $\leadsto$ Result | Test $\leadsto$ Result |
|---|---|---|---|---|---|
| No-$X$ | 96 | 0 | 8 | $X \leadsto -$ | $XB \leadsto CR$ |
| Few-$X$ | 96 | 4 | 8 | $X \leadsto CR$ | $XB \leadsto CR$ |
| Many-$X$ | 96 | 48 | 8 | $X \leadsto -$ | $XB \leadsto -$ |

Table 1: A summary of some of the experiments of Yin et al. [11]. The $US$ was a footshock; $A$ = white noise or buzzer sound; $X$ = tone; $B$ = click train.

## 3  Second-Order Conditioning and Conditioned Inhibition

We use the model to shed light on the relationship between two classical conditioning phenomena, second-order conditioning and conditioned inhibition. The procedures for establishing a second-order excitor and a conditioned inhibitor are similar, yet the results are drastically different. Both procedures involve two kinds of trials: a conditioned stimulus $A$ is presented with the $US$ ($A$-$US$); and $A$ is also presented with a target conditioned stimulus $X$ in unreinforced trials ($A$-$X$). In second order conditioning, $X$ becomes an excitor — it is associated with increased probability of reinforcement, demonstrated by conditioned responding. But in conditioned inhibition, $X$ becomes an inhibitor, i.e. associated with decreased probability of reinforcement. Inhibition is probed with two tests: a *transfer test*, in which the inhibitor is paired with a second excitor $B$ and shown to reduce conditioned responding, and a *retardation test*, in which the time course of response development under subsequent excitatory $X$-$US$ training is retarded relative to naive animals.

Yin et al. [11] explored the dimensions of these two procedures in an effort to distill the essential requirements for each. Under previous theories [8], it might have seemed that the crucial distinction between second order conditioning and conditioned inhibition had to do with either blocked versus interspersed trials, or with sequential versus simultaneous presentation of the $CS$es. However, they found that using only interspersed trials and simultaneous presentation of the conditioned stimuli, they were able to shift from second-order conditioning to conditioned inhibition simply by increasing the number of $A$-$X$ pairings.[2] Table 1 summarizes the relevant details of the experiment.

From a theoretical perspective, these results present a challenge for models of conditioning. Why do animals so drastically change their behavior regarding $X$ given only more of the same kind of $A$-$X$ experience? Bayesian model averaging offers some insight.

We simulated the experiments of Yin et al., matching their numbers for each type of trial, as shown in Table 1. Results of the MCMC approximation of the Bayesian model average integration are shown in Figure 2. All MCMC runs were at least $5 \times 10^6$ iterations long excluding a *burn-in* of $1 \times 10^6$ iterations. The sequences were subsampled to $2.5 \times 10^4$.

In Figure 2a, we see that $P(US \mid X, \mathcal{D})$ reveals significant second order conditioning with few $A$-$X$ trials. With more trials the predicted probability of reinforcement quickly decreases. These results are consistent with the findings of Yin et al., as shown in Table 1. With few $A$-$X$ trials there are insufficient data to justify a complicated model that accurately fits the data. Due to the automatic Occam's razor and the prior preference for simple models, high posterior density is inferred for the simple model of Figure 3a. This model combines the stimuli from all trial types and attributes them to a single latent cause. When $X$ is tested alone, its connection to the $US$ through the latent cause results in a large $P(US \mid X, \mathcal{D})$.

With more training trials, the preference for simpler models is more successfully offset and more complicated models — capable of describing the data more accurately — are given

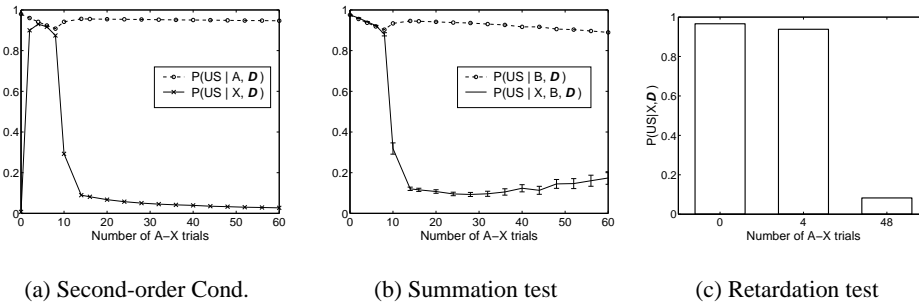

|(a) Second-order Cond.|(b) Summation test|(c) Retardation test|

Figure 2: A summary of the simulation results. Error bars indicate the $3\sigma$ margin in the standard error of the estimate (we omit very small error bars). (a) $P(US \mid X, \mathcal{D})$ and $P(US \mid A, \mathcal{D})$ as a function of $A$-$X$ trials. For few trials (2 to 8), $P(US \mid X, \mathcal{D})$ is high, indicative of second-order conditioning. (b) $P(US \mid X, B, \mathcal{D})$ and $P(US \mid B, \mathcal{D})$ as a function of number of $A$-$X$ trials. After 10 trials, $X$ is able to significantly reduce the predicted probability of reinforcement generated by the presentation of $B$. (c) Results of a retardation test. With many $A$-$X$ trials, acquisition of an excitatory association to $X$ is retarded.

greater posterior density (Figure 3c). An example of such a model is shown in Figure 3b. In the model, $X$ is made a conditioned inhibitor by a negative valued weight between $x_2$ and $X$. In testing $X$ with a transfer excitor $B$, as shown in Figure 2, this weight acts to cancel a positive correlation between $B$ and the $US$. Note that the shift from excitation to inhibition is due to inclusion of uncertainty over models; inferring the parameters with the more complex model fixed would result in immediate inhibition. In their experiment, Yin et al. also conducted a retardation test of conditioned inhibition for $X$. We follow their procedure and include in $\mathcal{D}$ 3 $X$-$US$ trials. Our retardation test results are shown in Figure 2 and are in agreement with the findings of Yin et al.

A further mystery about conditioned inhibitors, from the perspective of the benchmark theory of Rescorla and Wagner [7], is the nonextinction effect: repeated presentations of a conditioned inhibitor $X$ alone and unreinforced do not extinguish its inhibitory properties. An experiment by Williams and Overmier [10] demonstrated that unpaired presentations of a conditioned inhibitor can actually enhance its ability to suppress responding in a transfer test. Our model shows the same effect, as illustrated with a dramatic test in Figure 4. Here we used the previous dataset with only 8 $A$-$X$ pairings and added a number of unpaired presentations of $X$. The additional unpaired presentations shift the model from a second-order conditioning regime to a conditioned inhibition regime. The extinction trials suppress posterior density over simple models that exhibit a positive correlation between $X$ and $US$, shifting density to more complex models and unmasking the inhibitor.

## 4   Discussion

We have demonstrated our ideas in the context of a very abstract set of candidate models, ignoring the temporal arrangement of trials and of the events within them. Obviously, both of these issues have important effects, and the present framework can be straightforwardly generalized to account for them, with the addition of temporal dependencies to the latent variables [1] and the removal of the stationarity assumption [4].

An odd but key concept in early models of classical conditioning is the "configural unit," a detector for a conjunction of co-active stimuli. "Configural learning" theories (e.g. [6])

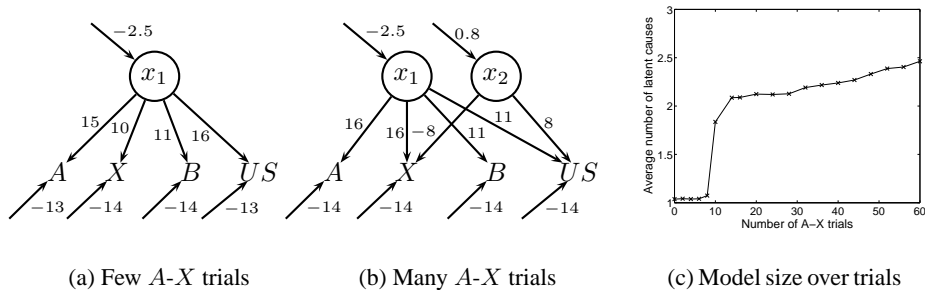

(a) Few $A$-$X$ trials      (b) Many $A$-$X$ trials      (c) Model size over trials

Figure 3: Sigmoid belief networks with high probability density under the posterior. (a) After a few $A$-$X$ pairings: this model exhibits second-order conditioning. (b) After many $A$-$X$ pairings: this model exhibits conditioned inhibition. (c) The average number of latent causes as a function of $A$-$X$ pairings.

rely on heuristics for creating such units in response to observations, a rough-and-ready sort of model structure learning. With a stimulus configuration represented through a latent cause, our theory provides a clearer prescription for how to reason about model structure. Our framework can be applied to a reservoir of configural learning experiments, including negative and positive patterning and a host of others. Another body of data on which our work may shed light is acquisition of a conditioned response. Recent theories of acquisition (e.g. [4]) propose that animals respond to a conditioned stimulus ($CS$) when the difference in the reinforcement rate between the presence and absence of the $CS$ satisfies some test of significance. From the perspective of our model, this test looks like a heuristic for choosing between generative models of stimulus delivery that differ as to whether the $CS$ and $US$ are correlated through a shared hidden cause.

To our knowledge, the relationship between second-order conditioning and conditioned inhibition has never been explicitly studied using previous theories. This is in part because the majority of classical conditioning theories do not account for second-order conditioning at all, since they typically consider learning only about $CS$-$US$ but not $CS$-$CS$ correlations. Models based on temporal difference learning [8] predict second-order conditioning, but only if the two $CS$es are presented sequentially (not true of the experiment considered here). Second-order conditioning can also be predicted if the $A$-$X$ pairings cause some sort of representational change so that $A$'s excitatory associations generalize to $X$. Yin et al. [11] suggest that if this representational learning is fast (as in [6], though that theory would need to be modified to include any second-order effects) and if conditioned inhibition accrues only gradually by error-driven learning [7], then second-order conditioning will dominate initially. The details of such an account seem never to have been worked out, and even if they were, such a mechanistic theory would be considerably less illuminating than our theory as to the normative reasons *why* the animals should predict as they do.

**Acknowledgments**

This work was supported by National Science Foundation grants IIS-9978403 and DGE-9987588, and by AFRL contract F30602–01–C–0219, DARPA's MICA program. We thank Peter Dayan and Maneesh Sahani for helpful discussions.

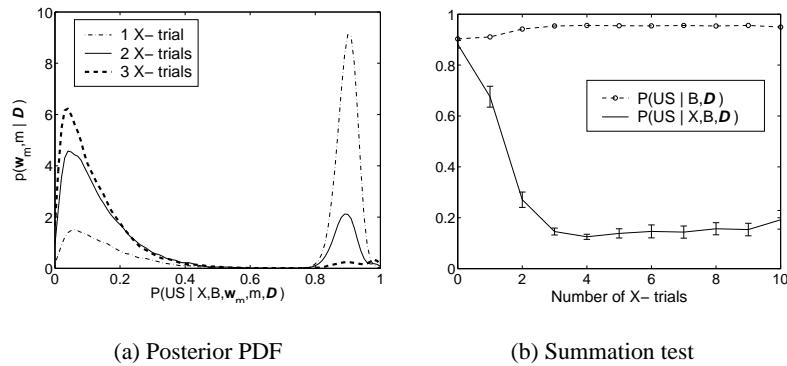

(a) Posterior PDF          (b) Summation test

Figure 4: Effect of adding unpaired presentations of $X$ on the strength of $X$ as an inhibitor. (a) Posterior probability of models which predict different values of $P(US \mid X, B)$. With only 1 unpaired presentation of $X$, most models predict a high probability of $US$ (second-order conditioning). With 2 or 3 unpaired presentations of $X$, models which predict a low $P(US \mid X, B)$ get more posterior weight (conditioned inhibition). (b) A plot contrasting $P(US \mid B, \mathcal{D})$ and $P(US \mid X, B, \mathcal{D})$ as a function of unpaired $X$ trials. The reduction in the probability of reinforcement indicates an enhancement of the inhibitory strength of $X$. Error bars indicate the $3\sigma$ margin in the standard error in the estimate (omitting small error bars).

## Footnotes

[1]The proposal acceptance probability satisfies detailed balance for each type of jump.

[2]In other conditions, trial ordering was shown to have an additional effect; this is outside the scope of the present theory due to our stationarity assumptions.

# References

[1] A. C. Courville and D. S. Touretzky. Modeling temporal structure in classical conditioning. In *Advances in Neural Information Processing Systems 14*, pages 3–10, Cambridge, MA, 2002. MIT Press.

[2] P. J. Green. Reversible jump Markov chain Monte Carlo computation and Bayesian model determination. *Biometrika*, 82:711–732, 1995.

[3] Y. Iba. Extended ensemble Monte Carlo. *International Journal of Modern Physics C*, 12(5):623–656, 2001.

[4] S. Kakade and P. Dayan. Acquisition and extinction in autoshaping. *Psychological Review*, 109:533–544, 2002.

[5] D. J. C. MacKay. Bayesian model comparison and backprop nets. In *Advances in Neural Information Processing Systems 4*, Cambridge, MA, 1991. MIT Press.

[6] J. M. Pearce. Similarity and discrimination: A selective review and a connectionist model. *Psychological Review*, 101:587–607, 1994.

[7] R. A. Rescorla and A. R. Wagner. A theory of Pavlovian conditioning: Variations in the effectiveness of reinforcement and nonreinforcement. In A. H. Black and W. F. Prokasy, editors, *Classical Conditioning II*. Appleton-Century-Crofts, 1972.

[8] R. S. Sutton and A. G. Barto. Time-derivative models of Pavlovian reinforcement. In M. Gabriel and J. Moore, editors, *Learning and Computational Neuroscience: Foundations of Adaptive Networks*, chapter 12, pages 497–537. MIT Press, 1990.

[9] J. Tenenbaum and T. Griffiths. Structure learning in human causal induction. In *Advances in Neural Information Processing Systems 13*, pages 59–65, Cambridge, MA, 2001. MIT Press.

[10] D. A. Williams and J. B. Overmier. Some types of conditioned inhibitors carry collateral excitatory associations. *Learning and Motivation*, 19:345–368, 1988.

[11] H. Yin, R. C. Barnet, and R. R. Miller. Second-order conditioning and Pavlovian conditioned inhibition: Operational similarities and differences. *Journal of Experimental Psychology: Animal Behavior Processes*, 20(4):419–428, 1994.
